# Estimating vector fields using sparse basis field expansions

**Stefan Haufe**[1, 2, *]    **Vadim V. Nikulin**[3, 4]    **Andreas Ziehe**[1, 2]    **Klaus-Robert Müller**[1, 2, 4]

**Guido Nolte**[2]

[1]TU Berlin, Dept. of Computer Science, Machine Learning Laboratory, Berlin, Germany
[2]Fraunhofer Institute FIRST (IDA), Berlin, Germany
[3]Charité University Medicine, Dept. of Neurology, Campus Benjamin Franklin, Berlin, Germany
[4]Bernstein Center for Computational Neuroscience, Berlin, Germany
[*] `haufe@cs.tu-berlin.de`

## Abstract

We introduce a novel framework for estimating vector fields using sparse basis field expansions (S-FLEX). The notion of basis fields, which are an extension of scalar basis functions, arises naturally in our framework from a *rotational invariance* requirement. We consider a regression setting as well as inverse problems. All variants discussed lead to *second-order cone programming* formulations. While our framework is generally applicable to any type of vector field, we focus in this paper on applying it to solving the EEG/MEG inverse problem. It is shown that significantly more precise and neurophysiologically more plausible location and shape estimates of cerebral current sources from EEG/MEG measurements become possible with our method when comparing to the state-of-the-art.

## 1  Introduction

Current machine learning is frequently concerned with the estimation of functions with multivariate output. While in many cases the outputs can be treated as mere collections of scalars (e.g. different color channels in image processing), in some contexts there might be a deeper interpretation of them as *spatial* vectors with a *direction* and a *magnitude*. Such "truly" vectorial functions are called *vector fields* and become manifest for example in optical flow fields, electromagnetic fields and wind fields in meteorology. Vector field estimators have to take into account that the numerical representation of a vector depends on the coordinate system it is measured in. That is, the estimate should be *invariant* with respect to a rotation of the coordinate system.

Let $\mathbf{v} : \mathbb{R}^P \mapsto \mathbb{R}^Q$ be a vector field. Mathematically speaking, we are seeking to approximate $\mathbf{v}$ by a field $\hat{\mathbf{v}}$ using empirical measurements. Here we consider two types of measurements. The first type are direct samples $(\mathbf{x}_n, \mathbf{y}_n), \mathbf{x}_n \in \mathbb{R}^P, \mathbf{y}_n \in \mathbb{R}^Q, n = 1, \ldots, N$ of $\mathbf{v}$ leading to a *regression problem*. The second case occurs, if only indirect measurements $z_m \in \mathbb{R}, m = 1, \ldots, M$ are available, which we assume to be generated by a known linear[1] transformation of the vector field outputs $\mathbf{y}_n$ belonging to nodes $\mathbf{x}_n, n = 1, \ldots, N$. This kind of estimation problem is known as an *inverse problem*. Let $\mathbf{z} = (z_1, \ldots, z_M)^T$ denote the vector of indirect measurements, $Y = (\mathbf{y}_1^T, \ldots, \mathbf{y}_N^T)^T$ the $N \times Q$ matrix of vector field outputs and $\mathbf{vec}(Y)$ a column vector containing the stacked transposed rows of $Y$. The linear relationship between $Y$ and $\mathbf{z}$ can be written as $\mathbf{z} = F \mathbf{vec}(Y)$ using the *forward model* $F \in \mathbb{R}^{M \times NQ}$.

As an example of an inverse problem consider the way humans localize acoustic sources. Here $\mathbf{z}$ comprises the signal arriving at the ears, $\mathbf{v}$ is the spatial distribution of the sound sources and $F$ is given by physical equations of sound propagation. Using information from two ears, humans do already very well in estimating the direction of incoming sounds. By further incorporating prior knowledge, e.g. on the loudness of the sources, $\mathbf{v}$ can usually be well approximated. The use of prior knowledge (a.k.a. *regularization*) is indeed the most effective strategy for solving inverse problems [13], which are inherently ambiguous. Hence, the same mechanisms used to avoid overfitting in, e.g., regression may be applied to cope with the ambiguity of inverse problems.

For the estimation of scalar functions, methods that utilize sparse linear combinations of *basis functions* have gained considerable attention recently (e.g. the "lasso" [14]). Apart from the computational tractability that comes with the sparsity of the learned model, the possibility of interpreting the estimates in terms of their basis functions is a particularly appealing feature of these methods. While sparse expansions are also desirable in vector field estimation, lasso and similar methods cannot be used for that purpose, as they break rotational invariance in the output space $\mathbb{R}^Q$. This is easily seen as sparse methods tend to select different basis functions in each of the $Q$ dimensions.

Only few attempts have been made on rotation-invariant sparse vector field expansions so far. In [8] a dense expansion is discussed, which could be modified to a sparse version maintaining rotational invariance. Unfortunately, this method is restricted to approximating *curl-free* fields. In contrast, we here propose a method that can be used to decompose any vector field. We will derive the general framework in section 2. In section 3 we will apply the (appropriately customized) method for solving the EEG/MEG inverse problem. Finally, we will draw a brief conclusion in section 4.

## 2 Method

Our model is based on the assumption that $\mathbf{v}$ can be well approximated by a linear combination of some *basis fields*. A basis field is defined here (unlike in [8]) as a vector field, in which all output vectors point in the same direction, while the magnitudes are proportional to a scalar (basis) function $b : \mathbb{R}^P \mapsto \mathbb{R}$. As demonstrated in Fig. 1, this model has an expressive power which is comparable to a basis function expansion of scalar functions. Given a set (*dictionary*) of basis functions $b_l(\mathbf{x}), l = 1, \ldots, L$, the basis field expansion is written as

$$\mathbf{v}(\mathbf{x}) = \sum_{l=1}^{L} \mathbf{c}_l b_l(\mathbf{x}) , \tag{1}$$

with coefficients $\mathbf{c}_l \in \mathbb{R}^Q, l = 1, \ldots, L$ to be estimated. Note that by including one coefficient for each output dimension, both orientations and proportionality factors are learned in this model (the term "basis field" thus refers to a basis function with learned coefficients). In order to select a small set of fields, most of the coefficient vectors $\mathbf{c}_l$ have to vanish. This can be accomplished by solving a least-squares problem with an additional lasso-like $\ell_1$-norm penalty on the coefficients. However, care has to be taken in order to maintain rotational invariance of the solution. We here propose to use a regularizer that imposes sparsity *and* is invariant with respect to rotations, namely the $\ell_1$-norm of the magnitudes of the coefficient vectors. Let $C = (\mathbf{c}_1, \ldots, \mathbf{c}_L)^T \in \mathbb{R}^{L \times Q}$ contain the coefficients and

$$B = \begin{pmatrix} b_1(\mathbf{x}_1) & \ldots & b_L(\mathbf{x}_1) \\ \vdots & & \vdots \\ b_1(\mathbf{x}_N) & \ldots & b_L(\mathbf{x}_N) \end{pmatrix} \in \mathbb{R}^{N \times L} \tag{2}$$

the basis functions evaluated at the $\mathbf{x}_n$. The parameters are estimated using

$$\hat{C} = \arg \min_C \ \mathcal{L}(C) + \lambda \mathcal{R}(C) , \tag{3}$$

where $\mathcal{R}(C) = \|C\|_{1,2} = \sum_{l=1}^{L} \|\mathbf{c}_l\|_2$ is the regularizer (the so-called $\ell_{1,2}$-norm of the matrix $C$), $\mathcal{L}(C)$ is the quadratic loss function, which is defined by $\mathcal{L}(C) = \|\operatorname{vec}(Y - BC)\|_2^2$ in the regression case and $\mathcal{L}(C) = \|\mathbf{z} - F \operatorname{vec}(BC)\|_2^2$ in the inverse reconstruction case, and $\lambda$ is a positive constant.

In the statistics literature $\ell_{1,2}$-norm regularization is already known as a general mechanism for achieving sparsity of grouped predictors [18]. Besides vector field estimation, this concept has natural applications in, e.g, multiple kernel learning [1] and channel selection for brain computer interfacing [15]. It has also recently been considered in the general multiple output setting [17].

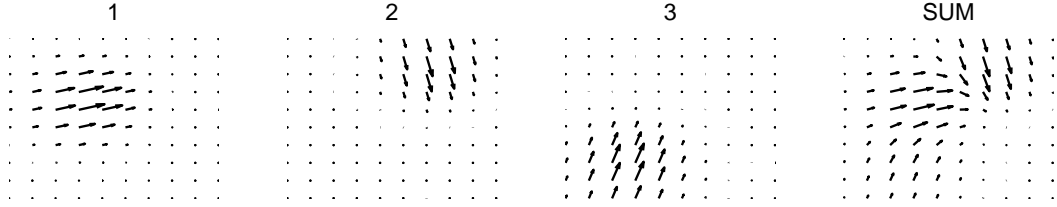

Figure 1: Complicated vector field (SUM) as a sum of three basis fields (1-3).

## 2.1 Rotational Invariance

Rotational invariance, in the sense that the estimates after rotation of the coordinates axes are equal to the rotated estimates, is a desirable property of an estimator. One has to distinguish invariance in input- from invariance in output space. The former requirement may arise in many estimation settings and can be fulfilled by the choice of appropriate basis functions $b_l(\mathbf{x})$. The latter one is specific to vector field estimation and has to be assured by formulating a rotationally invariant cost function. Our proposed estimator Eq. 3 is rotationally invariant. This is due to the use of the $\ell_2$-norm in output space $\mathbb{R}^Q$, which does not change under rotation. I.e. for an orthogonal matrix $R \in \mathbb{R}^{Q \times Q}, R^T R = I$

$$\sum_{l=1}^{L} \|R\mathbf{c}_l\|_2 = \sum_{l=1}^{L} \sqrt{\operatorname{tr}(\mathbf{c}_l^T R^T R \mathbf{c}_l)} = \sum_{l=1}^{L} \|\mathbf{c}_l\|_2 \ . \tag{4}$$

For the same argument, additional regularizers $\mathcal{R}_*(C) = \|\operatorname{\mathbf{vec}}(D_* C)\|_2^2$ (the well-known Tikhonov regularizer) or $\mathcal{R}_+(C) = \|D_+ C\|_{1,2}$ (promoting sparsity of the linearly transformed vectors) may be introduced without breaking the rotational invariance in $\mathbb{R}^Q$.

## 2.2 Optimization

Eq. 3 is a convex problem, composed of the quadratic term $\mathcal{L}(C)$ and the convex nondifferentiable term $\mathcal{R}(C)$. It is equivalent to the following program

$$\hat{C} \quad = \quad \arg \quad \min_{C, \mathbf{u}} \quad \sum_{l=1}^{L} u_l \quad \text{s.t.} \quad \|\mathbf{c}_l\|_2 \quad \leq \quad u_l \ , \quad l = 1, \ldots, L$$
$$\mathcal{L}(C) \quad \leq \quad \varepsilon \ , \tag{5}$$

in which a linear function of the variables is minimized subject to quadratic and second-order cone constraints [6]. The latter constraints are obtained by introducing auxiliary variables $u_l \in \mathbb{R}, l = 1, \ldots, L$ encoding upper bounds of the magnitudes of the coefficient vectors. Problem Eq. 5 is an instance of second-order cone programming (SOCP), a standard class of convex programs, for which efficient interior-point based solvers are available. The problem stays inside the SOCP class even if the original formulation is modified in any of the following ways:

- Additional regularizers $\mathcal{R}_+(C)$ or $\mathcal{R}_*(C)$ are used.

- The quadratic loss function is replaced by a more robust $\ell_1$-norm based loss (e.g. hinge loss). In the regression case, this loss should be defined based on the magnitude of the residual vector, which leads to a formulation involving the $\ell_{1,2}$-norm (and thus additional SOCP constraints).

- Complex basis functions (e.g. Fourier bases or Morlet wavelets) are used. This approach also requires complex coefficients, by which it is then possible not only to optimally scale the basis functions, but also to optimally shift their phase. Similarly, it is possible to reconstruct complex vector fields from complex measurements using real-valued basis functions.

# 3 Application to the EEG/MEG inverse problem

Vector fields occur, for example, in form of electrical currents in the brain, which are produced by postsynaptic neuronal processes. Knowledge of the electrical fields during a certain experimental condition allows one to draw conclusions about the locations in which the cognitive processing takes place and is thus of high value for research and medical diagnosis. Invasive measurements allow very local assessment of neuronal activations, but such procedure in humans is only possible when electrodes are implanted for treatment/diagnosis of neurological diseases, e.g., epilepsy. In the majority of cases recordings of cortical activity are performed with non-invasive measures such as electro- and magnetoencephalography, EEG and MEG respectively. The reconstruction of the current density from such measurements is an inverse problem.

## 3.1 Method specification

In the following the task is to infer the generating cerebral current density given an EEG measurement $\mathbf{z} \in \mathbb{R}^M$. The current density is a vector field $\mathbf{v} : \mathbb{R}^3 \mapsto \mathbb{R}^3$ assigning a vectorial current source to each location in the brain. We obtained a realistic head model from high-resolution MRI (magnetic resonance imaging) slices of a human head [4]. Inside the brain, we arranged 2142 nodes in a regular grid of 1 cm distance. The forward mapping $F \in \mathbb{R}^{M \times 2142 \cdot 3}$ from these nodes to the electrodes was constructed according to [9] – taking into account the realistic geometry and conductive properties of brain, skull and skin.

### Dictionary

In most applications the "true" sources are expected to be small in number and spatial extent. However, many commonly used methods estimate sources that almost cover the whole brain (e.g. [11]). Another group of methods delivers source estimates that are spatially sparse, but usually not rotationally invariant (e.g. [7]). Here often too many sources, which are scattered around the true sources, are estimated. Both the very smooth and the very sparse estimates are unrealistic from a physiological point of view. Only very recently, approaches capable of achieving a compromise between these two extremes have been outlined [16, 3]. For achieving a similar effect we here propose a sparse basis field expansion using radial basis functions. More specifically we consider spherical Gaussians

$$b_{n,s}(\mathbf{x}) = (2\pi\sigma_s)^{-\frac{3}{2}} \exp\left(-\frac{1}{2}\|\mathbf{x}-\mathbf{x}_n\|_2^2 \sigma_s^{-2}\right) \tag{6}$$

$s = 1, \ldots, 4$, having spatial standard deviations $\sigma_1 = 0.5$ cm, $\sigma_2 = 1$ cm, $\sigma_3 = 1.5$ cm, $\sigma_4 = 2$ cm and being centered at nodes $\mathbf{x}_n, n = 1, \ldots, N$ (see Fig. 2 for examples). Using this redundant dictionary our expectation is that sources of different spatial extent can be reconstructed by selecting the appropriate basis functions. Unlike the approaches taken in [16, 3] this approach does not require an additional hyperparameter for controlling the tradeoff between sparsity and smoothness.

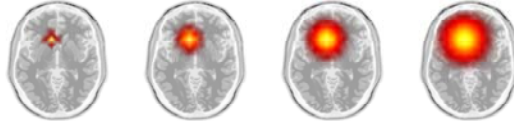

Figure 2: Gaussian basis functions with fixed center and standard deviations $0.5$ cm $- 2$ cm.

### Normalization

Our $\ell_{1,2}$-norm based regularization is a heuristic for selecting the smallest possible number of basis fields necessary to explain the measurement. Using this approach, however, not only the number of nonzero coefficient vectors, but also their magnitudes enter the cost function. It is therefore important to normalize the basis functions in order not to a-priori prefer some of them. Let $B_s$ be the $N \times N$ matrix containing the basis functions with standard deviation $\sigma_s$. The large matrix $B = (B_1/\|\operatorname{vec}(B_1)\|_1, \ldots, B_4/\|\operatorname{vec}(B_4)\|_1) \in \mathbb{R}^{N \times 4N}$ is then constructed using normalized $B_s$. By this means, no length scale is artificially prefered.

An estimation bias is also introduced by the location of the sources. Due to volume conduction, the signal captured at the sensors is much stronger for superficial sources compared to deep sources. In [10] the variance estimate $\hat{S} = \bar{F}^T \left( \bar{F} \bar{F}^T \right)^{-1} \bar{F} \in \mathbb{R}^{3N \times 3N}$ is derived for the (least-squares) estimated sources, where $\bar{F} = HF$ and $H = I - \mathbf{1}\mathbf{1}^T/\mathbf{1}^T\mathbf{1} \in \mathbb{R}^{M \times M}$. We found that $\hat{S}$ can be used for removing the location bias. This can be done by either penalizing activity at locations with high variance or by penalizing basis functions with high variance in the center. We here employ the former approach, as the latter may be problematic for basis functions with large extent. Using this approach, evaluation of $\hat{\mathbf{v}}(\mathbf{x})$ requires knowledge of the forward model for $\mathbf{x}$. Therefore, we restrict ourselves here to nodes $\mathbf{x}_n, n = 1, \dots, N$. Let $W_n \in \mathbb{R}^{3 \times 3}$ denote the inverse matrix square root of the part of $\hat{S}$ belonging to node $\mathbf{x}_n$. Defining

$$W = \begin{pmatrix} W_1 & \dots & 0 \\ \vdots & \ddots & \vdots \\ 0 & \dots & W_N \end{pmatrix} \in \mathbb{R}^{3N \times 3N} \ , \tag{7}$$

the coefficients are estimated using $\hat{C} = \arg\min_C \ \|C\|_{1,2}$ s.t. $\|\mathbf{z} - FW\,\mathbf{vec}(BC)\|_2^2 < \varepsilon$. The estimated current density at node $\mathbf{x}_n$ is $\hat{\mathbf{v}}(\mathbf{x}_n) = W_n \sum_{l=1}^L \hat{\mathbf{c}}_l b_l(\mathbf{x}_n)$.

## 3.2 Experiments

Validation of methods for inverse reconstruction is generally difficult due to the lack of a "ground truth". The measurements $\mathbf{z}$ cannot be used in this respect, as the main goal is not to predict the EEG/MEG measurements, but the vector field $\mathbf{v}(\mathbf{x})$ as accurately as possible. Therefore, the only way to evaluate inverse methods is to assess their ability to reconstruct known functions. We do this by reconstructing a) simulated current sources and b) sources of real EEG data that are already well-localized by other studies. For each EEG measurement, simulated or not, we conduct a $5 \times 5$ crossvalidation, i.e. we perform 25 inverse reconstructions based on different *training sets* containing 80 % of the electrodes. In each crossvalidation run, we evaluate two criteria. Most important is the *reconstruction error*, defined as $\mathcal{C}_\mathbf{y} = \|\,\mathbf{vec}(Y)/\|\,\mathbf{vec}(Y)\|_2 - \mathbf{vec}(\hat{Y}^{\text{tr}})/\|\,\mathbf{vec}(\hat{Y}^{\text{tr}})\|_2\|_2$, where $\hat{Y}^{\text{tr}}$ are the vector field outputs at nodes $\mathbf{x}_n, n = 1, \dots, N$ estimated using only the training set. This criterion can only be evaluated for the simulated data. For real and simulated data we also evaluate the *generalization error*, i.e. the error in the prediction of the remaining 20% (the *test set*) of the EEG measurements. This is defined as $\mathcal{C}_\mathbf{z} = \|\mathbf{z}^{\text{te}} - F^{\text{te}}\,\mathbf{vec}(\hat{Y}^{\text{tr}})\|_2^2$, where $\mathbf{z}^{\text{te}}$ and $F^{\text{te}}$ are the parts of $\mathbf{z}$ and $F$ belonging to the test set.

We compared the sparse basis field expansion (S-FLEX) approach using Gaussian basis functions (see section 3.1) to the commonly used approaches of LORETA [11] and Minimum Current Estimate (MCE) [7], and the recently proposed Focal Vectorfield Reconstruction (FVR) technique [3]. All three competitors correspond to using unit impulses as basis functions while employing different regularizers. The LORETA solution, e.g., is a Tikhonov regularized least-squares estimate while MCE is equivalent to applying lasso to each dimension separately, yielding current vectors that are biased towards being axes-parallel. We here used a variant of MCE, in which the original depth compensation approach was replaced by the approach outlined in section 3.1. Interestingly, FVR can be interpreted as a special case of S-FLEX employing the rotation-invariant regularizer $\mathcal{R}_+(C)$ to enforce both sparsity and smoothness. The tradeoff parameter $\alpha$ of this method was chosen as suggested in [3]. All methods were formulated such that the fitness of the solution was ensured by the constraint $\|\mathbf{z} - F\,\mathbf{vec}(\hat{Y}^{\text{tr}})\|_2^2 < \varepsilon$. The optimization was carried out using freely available packages for convex programming [12, 2].

**Simulated data**

We simulated current densities in the following way. First, we sampled outputs $\mathbf{y}_n, n = 1, \dots, N$ from a multivariate standard normal distribution. The function $(\mathbf{x}_n, \mathbf{y}_n)$ was then spatially smoothed using a Gaussian lowpass filter with standard deviation 2.5 cm. Finally, each $\mathbf{y}_n$ was shortened by the 90th percentile of the magnitudes of all $\mathbf{y}_n$ – leaving only 10% of the current vectors active. Current densities obtained by this procedure usually feature 2-3 active patches (sources) with small to medium extent and smoothly varying magnitude and orientation (see Fig. 3 for an example). This

behaviour was considered consistent with the general believe on the sources. We simulated five densities and computed respective pseudo-measurements for 118 channels using the forward model $F$. As no noise was injected in the system, $\varepsilon$ was set to zero in the following reconstruction.

**Real data**

We recorded 113-channel EEG of one healthy subject (male, 26 years) during electrical median nerve stimulation. The EEG electrodes were positioned according to the international 10-20 system. The exact positions were obtained using a 3D digitizer and mapped onto the surface of the head model. EEG data were recorded with sampling frequency of 2500 Hz and digitally bandpass-filtered between 15 Hz and 450 Hz. Left and right median nerves were stimulated in separate blocks by applying constant square 0.2 ms current pulses to the respective thenars. Current pulses had intensities above motor threshold (approx. 9 mA), inducing unintended twitches of the thumbs. The interstimulus interval varied randomly between 500 ms and 700 ms. About 1100 trials were recorded for each hand. Artifactual trials as well as artifactual electrodes were excluded from the analysis. For the remaining data, baseline correction was done based on the mean amplitude in the prestimulus interval (-100 ms to -10 ms). Finally, a single measurement vector was constructed by averaging the EEG amplitudes at 21 ms across 1946 trials (50% left hand, 50% right hand). By this means the EEG response to somatosensory input at the hands was captured with high signal-to-noise ratio (SNR). Based on that the brain areas representing left and right hand were to be reconstructed with $\varepsilon$ set according to the estimated SNR.

## 3.3 Results

Fig. 3 shows a simulated current density along with reconstructions according to LORETA, MCE, FVR and S-FLEX. From the figure it becomes apparent, that LORETA and MCE do not approximate the true current density very well. While the LORETA solution is rather blurry, merging the two true sources, the MCE solution exhibits many spikes, which could easily be misinterpreted as different sources. Note that the strong orientation bias of MCE cannot be seen in Fig. 3 as only dipole amplitudes are plotted. The estimates of FVR and S-FLEX approximately recover the shape of the sources. S-FLEX comes closest to the true shape, as its estimates are less focal than the ones of FVR. However, S-FLEX still slightly underestimate the extent of the sources.

The localization results of left and right N20 generators are shown in Fig. 4. The solutions of FVR and S-FLEX are almost indistinguishable. Both show activity concentrated in two major patches, one in each contralateral somatosensory cortex. This is in good agreement with the localization of the hand areas reported in the literature (e.g. [5]). LORETA estimates only one large active region over the whole central area, with the maximum lying exactly in between the hand areas. The MCE solution consists of eight spikes scattered across the whole somatosensory area.

Tab. 1 shows that S-FLEX generalizes better than its competitors, although insignificantly. More importantly S-FLEX outperforms its peers in terms of reconstruction accuracy. The distance to the runner-up FVR is, however, larger than expected from Fig. 3. This is due to the fact that the parameter of FVR controlling the tradeoff between sparsity and smoothness was fixed here to a value promoting "maximally sparse sources which are still smooth". While this might be a good assumption in practise, it was not rewarded in our validation setting. We here explicitly required reconstruction rather than shrinkage of the sources.

|  | $\mathcal{C}_{\mathbf{y}}$ SIM | $\mathcal{C}_{\mathbf{z}}$ SIM | $\mathcal{C}_{\mathbf{z}}$ REAL |
|---|---|---|---|
| LORETA | $1.00 \pm 0.01$ | $2.87 \pm 0.78$ | $8.18 \pm 1.38$ |
| FVR | $0.955 \pm 0.02$ | $1.21 \pm 1.00$ | $8.01 \pm 1.79$ |
| S-FLEX | $\mathbf{0.71} \pm 0.04$ | $0.952 \pm 0.28$ | $7.95 \pm 1.84$ |
| MCE | $1.21 \pm 0.01$ | $1.86 \pm 0.57$ | $8.13 \pm 1.60$ |

Table 1: Ability of LORETA, FVR, S-FLEX and MCE to reconstruct simulated currents ($\mathcal{C}_{\mathbf{y}}$ SIM) and generalization performance with respect to the EEG measurements ($\mathcal{C}_{\mathbf{z}}$ SIM/REAL). Winning entries (reaching significance) are shown in bold face.

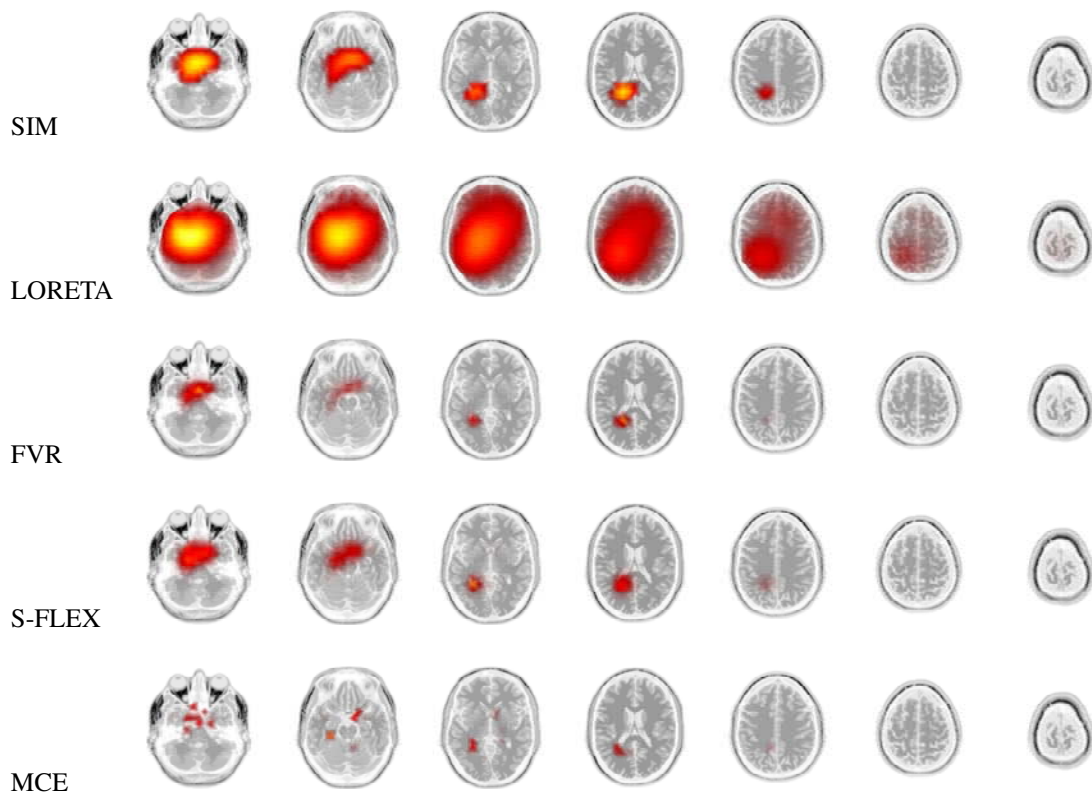

Figure 3: Simulated current density (SIM) and reconstruction according to LORETA, FVR, S-FLEX and MCE. Color encodes current magnitude.

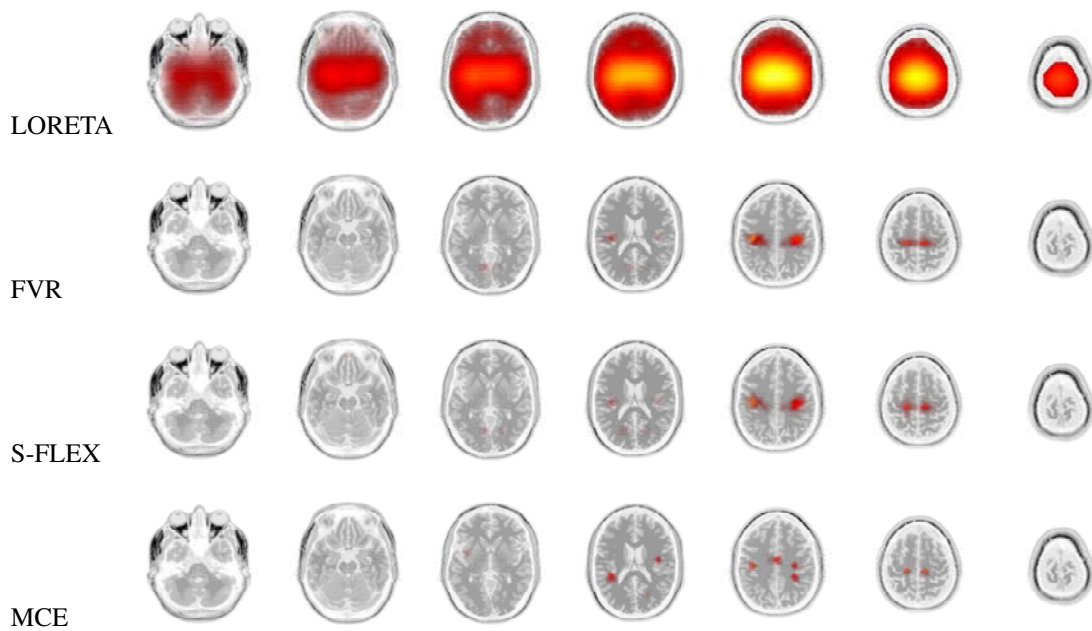

Figure 4: Localization of somatosensory evoked N20 generators according to LORETA, FVR, S-FLEX and MCE. Color encodes current magnitude.

## 4 Conclusion and Outlook

This paper contributes a novel and general methodology for obtaining sparse decompositions of vector fields. An important ingredient of our framework is the insight that the vector field estimate should be *invariant* with respect to a rotation of the coordinate system. Interestingly, the latter constraint together with sparsity leads to a *second-order cone programming* formulation.

We have focussed here on solving the EEG/MEG inverse problem, where our proposed S-FLEX approach outperformed the state-of-the-art in approximating the true shape of the current sources. However, other fields might as well benefit from the use of S-FLEX: in meteorology for example, an improved decomposition of wind fields into their driving components might provide novel insights that could be useful for better weather forecasting.

### Acknowledgments

This work was supported in part by the German BMBF grants BCCNB-A4 (FKZ 01GQ0415), BFNTB-A1 (FKZ 01GQ0850) and FaSor (FKZ 16SV2234). We thank Friederike Hohlefeld and Monika Weber for help in preparing the experiment, and Ryota Tomioka for fruitful discussions.

## Footnotes

[1]If the true relation is nonlinear, it is here assumed to be linearized.

## References

[1] F.R. Bach, G.R.G. Lanckriet, and M.I. Jordan. Multiple kernel learning, conic duality and the SMO algorithm. In *Proceedings of the Twenty-first International Conference on Machine Learning*, 2004.

[2] M. Grant, S. Boyd, and Y. Ye. CVX: Matlab Software for Disciplined Convex Programming, October 2006. http://www.stanford.edu/~boyd/cvx/, Version 1.0RC.

[3] S. Haufe, V.V. Nikulin, A. Ziehe, K.-R. Müller, and G. Nolte. Combining sparsity and rotational invariance in EEG/MEG source reconstruction. *NeuroImage*, 42(2):26–738, 2008.

[4] C.J. Holmes, R. Hoge, L. Collins, R. Woods, A.W. Toga, and A.C. Evans. Enhancement of MR images using registration for signal averaging. *J. Comput. Assist. Tomogr.*, 22(2):324–333, 1998.

[5] J. Huttunen, S. Komssi, and L. Lauronen. Spatial dynamics of population activities at S1 after median and ulnar nerve stimulation revisited: An MEG study. *NeuroImage*, 32:1024–1031, 2006.

[6] M.S. Lobo, L. Vandenberghe, S. Boyd, and H. Lebret. Applications of second-order cone programming. *Lin. Alg. Appl.*, 284:193–228, 1998.

[7] K. Matsuura and Y. Okabe. Selective minimum-norm solution of the biomagnetic inverse problem. *IEEE Trans. Biomed. Eng.*, 42:608–615, 1995.

[8] F.A. Mussa-Ivaldi. From basis functions to basis fields: vector field approximation from sparse data. *Biol. Cybern.*, 67:479–489, 1992.

[9] G. Nolte and G. Dassios. Analytic expansion of the EEG lead field for realistic volume conductors. *Phys. Med. Biol.*, 50:3807–3823, 2005.

[10] R.D. Pascual-Marqui. Standardized low-resolution brain electromagnetic tomography (sLORETA): technical details. *Meth. Find. Exp. Clin. Pharmacol.*, 24(1):5–12, 2002.

[11] R.D. Pascual-Marqui, C.M. Michel, and D. Lehmann. Low resolution electromagnetic tomography: a new method for localizing electrical activity in the brain. *Int. J. Psychophysiol.*, 18:49–65, 1994.

[12] J.F. Sturm. Using SeDuMi 1.02, a MATLAB toolbox for optimization over symmetric cones. *Optim. Method. Softw.*, 11–12:625–653, 1999.

[13] A. Tarantola. *Inverse Problem Theory and Model Parameter Estimation*. SIAM, Philadelphia, 2005.

[14] R. Tibshirani. Regression shrinkage and selection via the lasso. *J. Roy. Stat. Soc. B Meth.*, 58(1):267–288, 1996.

[15] R. Tomioka and S. Haufe. Combined classification and channel/basis selection with L1-L2 regularization with application to P300 speller system. In *Proceedings of the 4th International Brain-Computer Interface Workshop and Training Course 2008*. Verlag der Technischen Universität Graz, 2008.

[16] M. Vega-Hernández, E. Martínez-Montes, J.M. Sánchez-Bornot, A. Lage-Castellanos, and P.A. Valdés-Sosa. Penalized least squares methods for solving the EEG inverse problem. *Stat. Sinica*, 2008. In press.

[17] D.P. Wipf and B.D. Rao. An empirical bayesian strategy for solving the simultaneous sparse approximation problem. *IEEE Trans. Signal Proces.*, 55(7):3704–3716, 2007.

[18] M. Yuan and Y. Lin. Model selection and estimation in regression with grouped variables. *J. Roy. Stat. Soc. B Meth.*, 68(1):49–67, 2006.

